# Automatic Acquisition and Efficient Representation of Syntactic Structures

**Zach Solan, Eytan Ruppin, David Horn**
Faculty of Exact Sciences
Tel Aviv University
Tel Aviv, Israel 69978
{*rsolan,ruppin,horn*}*@post.tau.ac.il*

**Shimon Edelman**
Department of Psychology
Cornell University
Ithaca, NY 14853, USA
*se37@cornell.edu*

## Abstract

The distributional principle according to which morphemes that occur in identical contexts belong, in some sense, to the same category [1] has been advanced as a means for extracting syntactic structures from corpus data. We extend this principle by applying it recursively, and by using mutual information for estimating category coherence. The resulting model learns, in an unsupervised fashion, highly structured, distributed representations of syntactic knowledge from corpora. It also exhibits promising behavior in tasks usually thought to require representations anchored in a grammar, such as systematicity.

## 1   Motivation

Models dealing with the acquisition of syntactic knowledge are sharply divided into two classes, depending on whether they subscribe to some variant of the classical generative theory of syntax, or operate within the framework of "general-purpose" statistical or distributional learning. An example of the former is the model of [2], which attempts to learn syntactic structures such as Functional Category, as stipulated by the Government and Binding theory. An example of the latter model is Elman's widely used Simple Recursive Network (SRN) [3].

We believe that polarization between statistical and classical (generative, rule-based) approaches to syntax is counterproductive, because it hampers the integration of the stronger aspects of each method into a common powerful framework. Indeed, on the one hand, the statistical approach is geared to take advantage of the considerable progress made to date in the areas of distributed representation, probabilistic learning, and "connectionist" modeling. Yet, generic connectionist architectures are ill-suited to the abstraction and processing of symbolic information. On the other hand, classical rule-based systems excel in just those tasks, yet are brittle and difficult to train.

We present a scheme that acquires "raw" syntactic information construed in a distributional sense, yet also supports the distillation of rule-like regularities out of the accrued statistical knowledge. Our research is motivated by linguistic theories that postulate syntactic structures (and transformations) rooted in distributional data, as exemplified by the work of Zellig Harris [1].

## 2 The ADIOS model

The ADIOS (Automatic DIstillation Of Structure) model constructs syntactic representations of a sample of language from unlabeled corpus data. The model consists of two elements: (1) a Representational Data Structure (RDS) graph, and (2) a Pattern Acquisition (PA) algorithm that learns the RDS in an unsupervised fashion. The PA algorithm aims to detect *patterns* — repetitive sequences of 'significant' strings of primitives occurring in the corpus (Figure 1). In that, it is related to prior work on alignment-based learning [4] and regular expression ('local grammar') extraction [5] from corpora. We stress, however, that our algorithm requires no pre-judging either of the scope of the primitives or of their classification, say, into syntactic categories: all the information needed for its operation is extracted from the corpus in an unsupervised fashion.

In the initial phase of the PA algorithm the text is segmented down to the smallest possible morphological constituents (e.g., ed is split off both walked and bed; the algorithm later discovers that bed should be left whole, on statistical grounds).[1] This initial set of unique constituents is the vertex set of the newly formed RDS (multi-)graph. A directed edge is inserted between two vertices whenever the corresponding transition exists in the corpus (Figure 2(a)); the edge is labeled by the sentence number and by its within-sentence index. Thus, corpus sentences initially correspond to *paths* in the graph, a path being a sequence of edges that share the same sentence number.

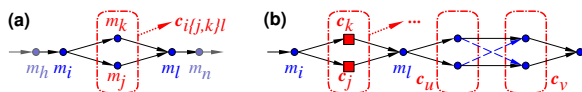

Figure 1: (a) Two sequences $m_i, m_j, m_l$ and $m_i, m_k, m_l$ form a pattern $c_{i\{j,k\}l} \doteq m_i, \{m_j, m_k\}, m_l$, which allows $m_j$ and $m_k$ to be attributed to the same equivalence class, following the principle of complementary distributions [1]. Both the length of the shared context and the cohesiveness of the equivalence class need to be taken into account in estimating the goodness of the candidate pattern (see eq. 1). (b) Patterns can serve as constituents in their own right; recursively abstracting patterns from a corpus allows us to capture the syntactic regularities concisely, yet expressively. Abstraction also supports generalization: in this schematic illustration, two new paths (dashed lines) emerge from the formation of equivalence classes associated with $c_u$ and $c_v$.

In the second phase, the PA algorithm repeatedly scans the RDS graph for Significant Patterns (sequences of constituents) (SP), which are then used to modify the graph (Algorithm 1). For each path $p_i$, the algorithm constructs a list of candidate constituents, $c_{i1}, \ldots, c_{ik}$. Each of these consists of a 'prefix' (sequence of graph edges), an equivalence class of vertices, and a 'suffix' (another sequence of edges; cf. Figure 2(b)).

The criterion $I'$ for judging pattern significance combines a syntagmatic consideration (the pattern must be long enough) with a paradigmatic one (its constituents $c_1, \ldots, c_k$ must have high mutual information):

$$I'(c_1, c_2, \ldots, c_k) = e^{-(L/k)^2} P(c_1, c_2, \ldots, c_k) \log \frac{P(c_1, c_2, \ldots, c_k)}{\Pi_{j=1}^k P(c_j)} \qquad (1)$$

where $L$ is the typical context length and $k$ is the length of the candidate pattern; the probabilities associated with a $c_j$ are estimated from frequencies that are immediately available

**Algorithm 1** PA (pattern acquisition), phase 2
---
1: **while** patterns exist **do**
2:   **for all** path ∈ graph **do** {path=sentence; graph=corpus}
3:     **for all** source_node ∈ path **do**
4:       **for all** sink_node ∈ path **do** {source and sink can be equivalence classes}
5:         degree_of_separation = path_index(sink) − path_index(source);
6:         pattern_table ⇐ detect_patterns(source, sink, degree_of_separation, equivalence_table);
7:       **end for**
8:     **end for**
9:     winner ⇐ get_most_significant_pattern(pattern_table);
10:     equivalence_table ⇐ detect_equivalences(graph, winner);
11:     graph ⇐ rewire_graph(graph, winner);
12:   **end for**
13: **end while**
---

in the graph (e.g., the out-degree of a node is related to the marginal probability of the corresponding $c_j$). Equation 1 balances two opposing "forces" in pattern formation: (1) the length of the pattern, and (2) the number and the cohesiveness of the set of examples that support it. On the one hand, shorter patterns are likely to be supported by more examples; on the other hand, they are also more likely to lead to over-generalization, because shorter patterns mean less context.

A pattern tagged as significant is added as a new vertex to the RDS graph, replacing the constituents and edges it subsumes (Figure 2). Note that only those edges of the multigraph that belong to the detected pattern are rewired; edges that belong to sequences not subsumed by the pattern are untouched. This highly context-sensitive approach to pattern abstraction, which is unique to our model, allows ADIOS to achieve a high degree of representational parsimony without sacrificing generalization power.

During the pass over the corpus the list of equivalence sets is updated continuously; the identification of new significant patterns is done using the *current* equivalence sets (Figure 3(d)). Thus, as the algorithm processes more and more text, it "bootstraps" itself and enriches the RDS graph structure with new SPs and their accompanying equivalence sets. The recursive nature of this process enables the algorithm to form more and more complex patterns, in a hierarchical manner. The relationships among these can be visualized recursively in a tree format, with tree depth corresponding to the level of recursion (e.g., Figure 3(c)). The PA algorithm halts if it processes a given amount of text without finding a new SP or equivalence set (in real-life language acquisition this process may never stop).

**Generalization.** A collection of patterns distilled from a corpus can be seen as an empirical grammar of sorts; cf. [6], p.63: 'the grammar of a language is simply an inventory of linguistic units." The patterns can eventually become highly abstract, thus endowing the model with an ability to generalize to unseen inputs. Generalization is possible, for example, when two equivalence classes are placed next to each other in a pattern, creating new paths among the members of the equivalence classes (dashed lines in Figure 1(b)). Generalization can also ensue from partial activation of existing patterns by novel inputs. This function is supported by the *input module*, designed to process a novel sentence by forming its distributed representation in terms of activities of existing patterns (Figure 6). These are computed by propagating activation from bottom (the terminals) to top (the patterns) of the RDS. The initial activities $w_j$ of the terminals $c_j$ are calculated given the novel input $s_1, \ldots, s_k$ as follows:

$$w_j = \max_{m=1..k} \{I(s_k, c_j)\} \qquad (2)$$

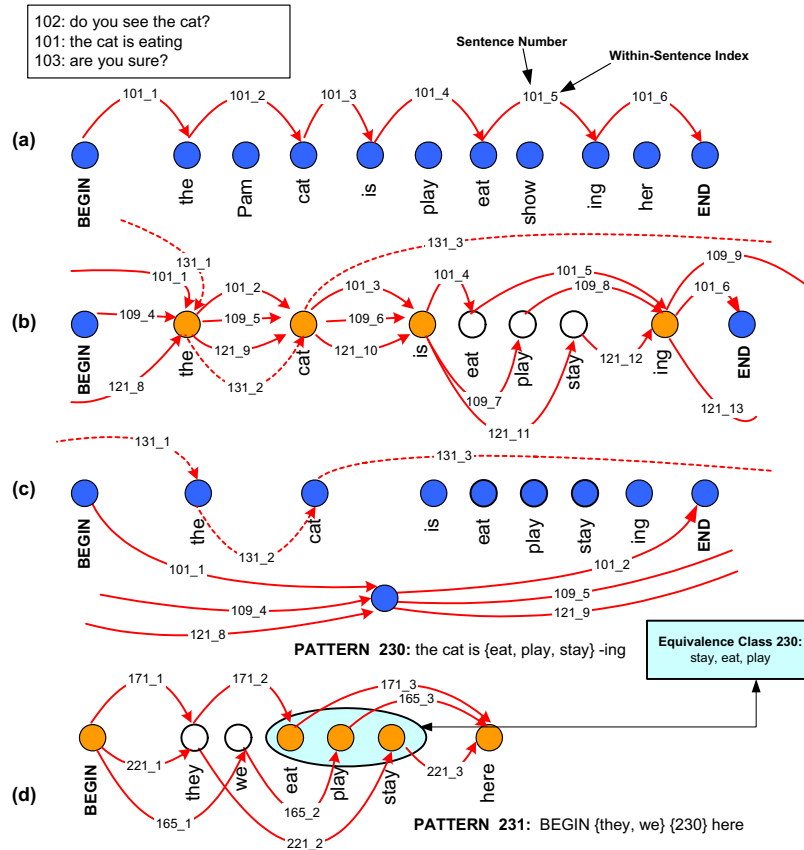

Figure 2: (a) A small portion of the RDS graph for a simple corpus, with sentence #101 (the cat is eat -ing) indicated by solid arcs. (b) This sentence joins a pattern the cat is {eat, play, stay} -ing, in which two others (#109,121) already participate. (c) The abstracted pattern, and the equivalence class associated with it (edges that belong to sequences not subsumed by this pattern, e.g., #131, are untouched). (d) The identification of new significant patterns is done using the acquired equivalence classes (e.g., #230). In this manner, the system "bootstraps" itself, recursively distilling more and more complex patterns.

where $I(s_k, c_j)$ is the mutual information between $s_k$ and $c_j$. For an equivalence class, the value propagated upwards is the strongest non-zero activation of its members; for a pattern, it is the average weight of the children nodes, on the condition that all the children were activated by adjacent inputs. Activity propagation continues until it reaches the top nodes of the pattern lattice. When the algorithm encounters a novel word, all the members of the terminal equivalence class contribute a value of $\epsilon$, which is then propagated upwards as usual. This enables the model to make an educated guess as to the meaning of the unfamiliar word, by considering the patterns that become active (Figure 6(b)).

# 3  Results

We now briefly describe the results of several studies designed to evaluate the viability of the ADIOS model, in which it was exposed to corpora of varying size and complexity.

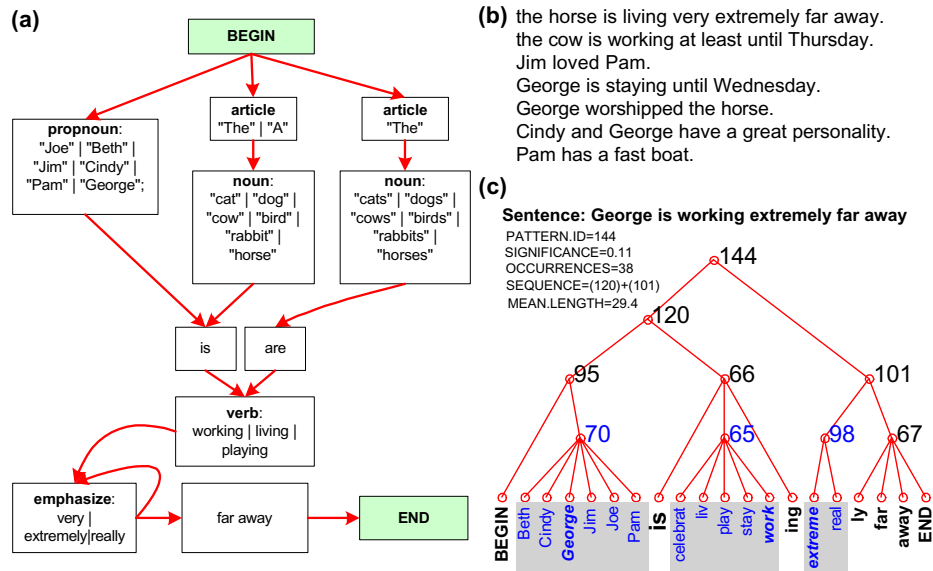

Figure 3: (a) A part of a simple grammar. (b) Some sentences generated by this grammar. (c) The structure of a sample sentence (pattern #144), presented in the form of a tree that captures the hierarchical relationships among constituents. Three equivalence classes are shown explicitly (highlighted).

**Emergence of syntactic structures.** Figure 3 shows an example of a sentence from a corpus produced by a simple artificial grammar and its ADIOS analysis (the use of a simple grammar, constructed with Rmutt, http://www.schneertz.com/rmutt, in these initial experiments allowed us to examine various properties of the model on tightly controlled data). The abstract representation of the sample sentence in Figure 3(c) looks very much like a parse tree, indicating that our method successfully identified the grammatical structure used to generate its data. To illustrate the gradual emergence of our model's ability for such concise representation of syntactic structures, we show in Figure 4, top, four trees built for the same sentence after exposing the model to progressively more data from the same corpus. Note that both the number of distinct patterns and the average number of patterns per sentence asymptote for this corpus after exposure to about 500 sentences (Figure 4, bottom).

**Novel inputs; systematicity.** An important characteristic of a cognitive representation scheme is its systematicity, measured by the ability to deal properly with structurally related items (see [7] for a definition and discussion). We have assessed the systematicity of the ADIOS model by splitting the corpus generated by the grammar of Figure 3 into training and test sets. After training the model on the former, we examined the representations of unseen sentences from the test set. A typical result appears in Figure 5; the general finding was of Level 3 systematicity according to the nomenclature of [7]. This example can be also understood using the concept of generating novel sentences from patterns, explained in detail below; the novel sentence (Beth is playing on Sunday) can be produced by the same pattern (#173) that accounts for the familiar sentence (the horse is playing on Thursday) that is a part of the training corpus.

The ADIOS system's input module allows it to process a novel sentence by forming its distributed representation in terms of activities of existing patterns. Figure 6 shows the activation of two patterns (#141 and #120) by a phrase that contains a word in a novel context (stay), as well as another word never before encountered in any context (5pm).

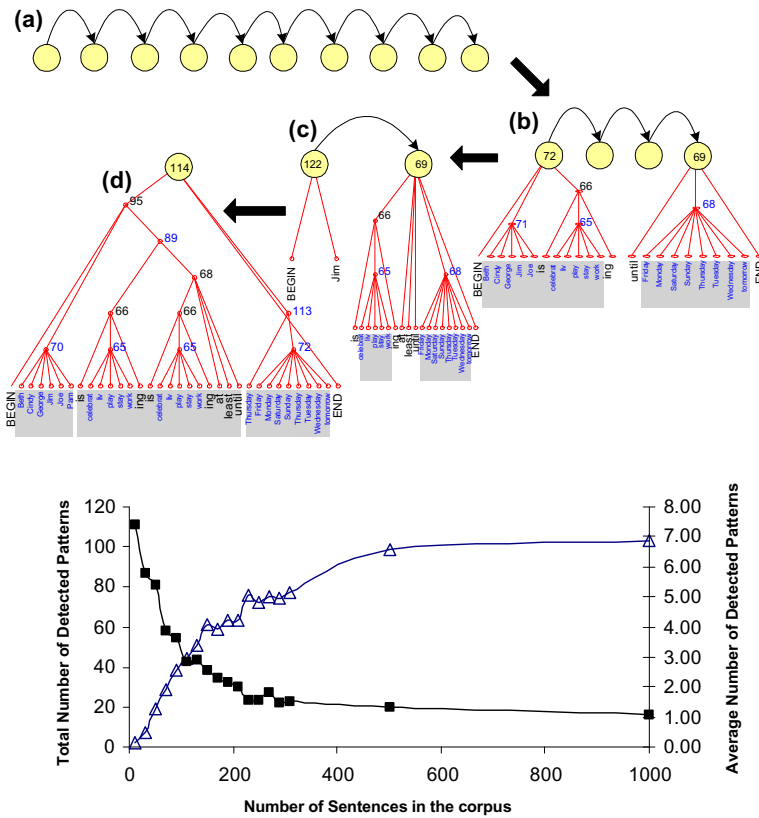

Figure 4: *Top:* the build-up of structured information with progressive exposure to a corpus generated by the simple grammar of Figure 3. (a) Prior to exposure. (b) 100 sentences. (c) 200 sentences. (d) 400 sentences. *Bottom:* the total number of detected patterns (△) and the average number of patterns in a sentence (■), plotted vs. corpus size.

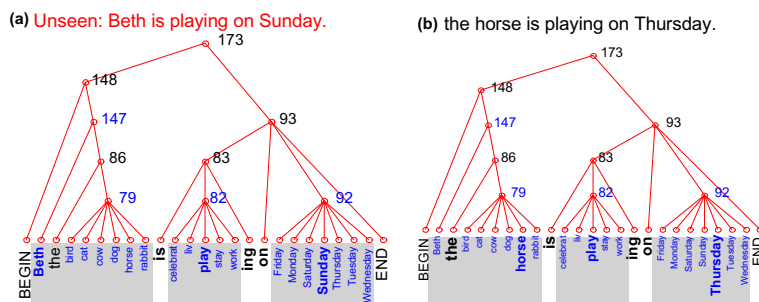

Figure 5: (a) Structured representation of an "unseen" sentence that had been excluded from the corpus used to learn the patterns; note that the detected structure is identical to that of (b), a "seen" sentence. The identity between the structures detected in (a) and (b) is a manifestation of Level-3 systematicity of the ADIOS model ("Novel Constituent: the test set contains at least one atomic constituent that did not appear anywhere in the training set"; see [7], pp.3-4).

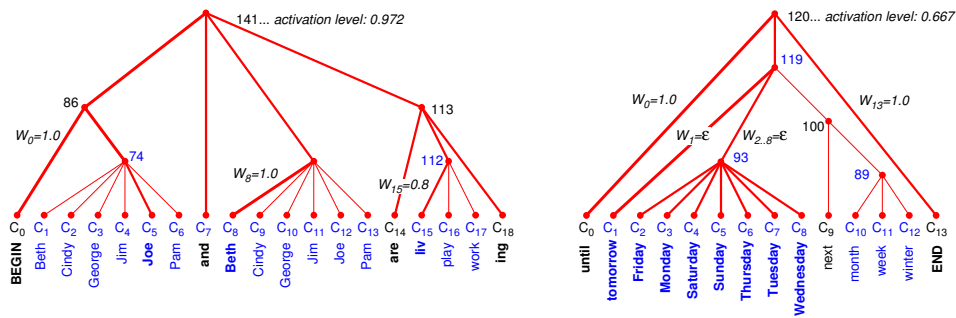

Figure 6: the input module in action (the two most relevant – highly active – patterns responding to the input Joe and Beth are staying until 5pm). Leaf activation is proportional to the mutual information between inputs and various members of the equivalence classes (e.g., on the left $W_{15} = 0.8$ is the mutual information between stay and liv, which is a member of equivalence class #112). It is then propagated upwards by taking the average at each junction.

**Working with real data: the CHILDES corpus.** To illustrate the scalability of our method, we describe here briefly the outcome of applying the PA algorithm to a subset of the CHILDES collection [8], which consists of transcribed speech produced by, or directed at, children. The corpus we selected contained 9665 sentences (74500 words) produced by parents. The results, one of which is shown in Figure 7, were encouraging: the algorithm found intuitively significant SPs and produced semantically adequate corresponding equivalence sets. Altogether, 1062 patterns and 775 equivalence classes were established. Representing the corpus in terms of these constituents resulted in a significant compression: the average number of constituents per sentence dropped from 6.70 in the raw data to 2.18 after training, and the entropy per letter was reduced from 2.6 to 1.5.

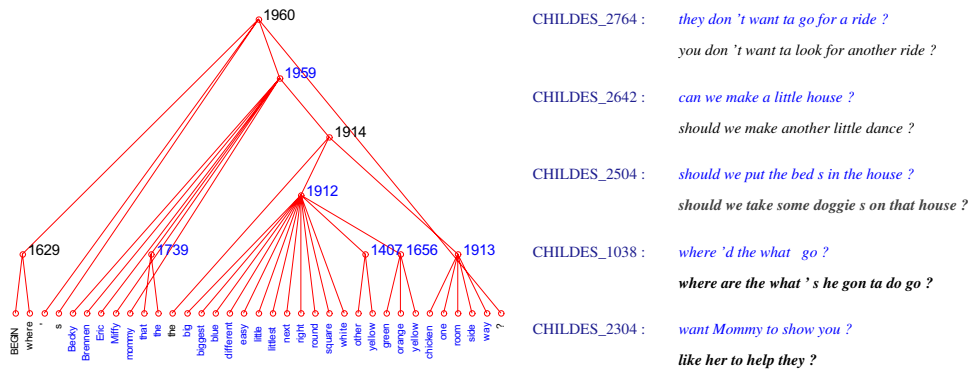

Figure 7: *Left:* a typical pattern extracted from a subset of the CHILDES corpora collection [8]. Hundreds of such patterns and equivalence classes (underscored in this figure) together constitute a concise representation of the raw data. Some of the phrases that can be described/generated by pattern 1960 are: where's the big room?; where's the yellow one?; where's Becky?; where's that?. *Right:* some of the phrases generated by ADIOS (lower lines in each pair) using sentences from CHILDES (upper lines) as examples. The generation module works by traversing the top-level pattern tree, stringing together lower-level patterns and selecting randomly one member from each equivalence class. Extensive testing (currently under way) is needed to determine whether the grammaticality of the newly generated phrases (which is at present less than ideal, as can be seen here) improves with more training data.

# 4 Concluding remarks

We have described a linguistic pattern acquisition algorithm that aims to achieve a streamlined representation by compactly representing recursively structured constituent patterns as single constituents, and by placing strings that have an identical backbone and similar context structure into the same equivalence class. Although our pattern-based representations may look like collections of finite automata, the information they contain is much richer, because of the recursive invocation of one pattern by another, and because of the context sensitivity implied by relationships among patterns. The sensitivity to context of pattern abstraction (during learning) and use (during generation) contributes greatly both to the conciseness of the ADIOS representation and to the conservative nature of its generative behavior. This context sensitivity — in particular, the manner whereby ADIOS balances syntagmatic and paradigmatic cues provided by the data — is mainly what distinguishes it from other current work on unsupervised probabilistic learning of syntax, such as [9, 10, 4].

In summary, finding a good set of structured units leads to the emergence of a convergent representation of language, which eventually changes less and less with progressive exposure to more data. The power of the constituent graph representation stems from the interacting ensembles of patterns and equivalence classes that comprise it. Together, the local patterns create global complexity and impose long-range order on the linguistic structures they encode. Some of the challenges implicit in this approach that we leave for future work are (1) interpreting the syntactic structures found by ADIOS in the context of contemporary theories of syntax, and (2) relating those structures to semantics.

*Acknowledgments.* We thank Regina Barzilai, Morten Christiansen, Dan Klein, Lillian Lee and Bo Pang for useful discussion and suggestions, and the US-Israel Binational Science Foundation, the Dan David Prize Foundation, the Adams Super Center for Brain Studies at TAU, and the Horowitz Center for Complexity Science for financial support.

## Footnotes

[1]We remark that the algorithm can work in any language, with any set of tokens, including individual characters – or phonemes, if applied to speech.

# References

[1] Z. S. Harris. Distributional structure. *Word*, 10:140–162, 1954.

[2] R. Kazman. Simulating the child's acquisition of the lexicon and syntax - experiences with Babel. *Machine Learning*, 16:87–120, 1994.

[3] J. L. Elman. Finding structure in time. *Cognitive Science*, 14:179–211, 1990.

[4] M. van Zaanen and P. Adriaans. Comparing two unsupervised grammar induction systems: Alignment-based learning vs. EMILE. Report 05, School of Computing, Leeds University, 2001.

[5] M. Gross. The construction of local grammars. In E. Roche and Y. Schabès, ed., *Finite-State Language Processing*, 329–354. MIT Press, Cambridge, MA, 1997.

[6] R. W. Langacker. *Foundations of cognitive grammar*, volume I: theoretical prerequisites. Stanford University Press, Stanford, CA, 1987.

[7] T. J. van Gelder and L. Niklasson. On being systematically connectionist. *Mind and Language*, 9:288–302, 1994.

[8] B. MacWhinney and C. Snow. The child language exchange system. *Journal of Computational Lingustics*, 12:271–296, 1985.

[9] D. Klein and C. D. Manning. Natural language grammar induction using a constituent-context model. In T. G. Dietterich, S. Becker, and Z. Ghahramani, ed., *Adv. in Neural Information Proc. Systems 14*, Cambridge, MA, 2002. MIT Press.

[10] A. Clark. *Unsupervised Language Acquisition: Theory and Practice*. PhD thesis, COGS, University of Sussex, 2001.
